# A Probabilistic Approach to Language Change

**Alexandre Bouchard-Côté**[*]    **Percy Liang**[*]    **Thomas L. Griffiths**[†]    **Dan Klein**[*]
[*]Computer Science Division    [†]Department of Psychology
University of California at Berkeley
Berkeley, CA 94720

## Abstract

We present a probabilistic approach to language change in which word forms are represented by phoneme sequences that undergo stochastic edits along the branches of a phylogenetic tree. This framework combines the advantages of the classical comparative method with the robustness of corpus-based probabilistic models. We use this framework to explore the consequences of two different schemes for defining probabilistic models of phonological change, evaluating these schemes by reconstructing ancient word forms of Romance languages. The result is an efficient inference procedure for automatically inferring ancient word forms from modern languages, which can be generalized to support inferences about linguistic phylogenies.

## 1   Introduction

Languages evolve over time, with words changing in form, meaning, and the ways in which they can be combined into sentences. Several centuries of linguistic analysis have shed light on some of the key properties of this evolutionary process, but many open questions remain. A classical example is the hypothetical *Proto-Indo-European* language, the reconstructed common ancestor of the modern Indo-European languages. While the existence and general characteristics of this proto-language are widely accepted, there is still debate regarding its precise phonology, the original homeland of its speakers, and the date of various events in its evolution. The study of how languages change over time is known as diachronic (or historical) linguistics (e.g., [4]).

Most of what we know about language change comes from the *comparative method*, in which words from different languages are compared in order to identify their relationships. The goal is to identify regular sound correspondences between languages and use these correspondences to infer the forms of proto-languages and the phylogenetic relationships between languages. The motivation for basing the analysis on sounds is that phonological changes are generally more systematic than syntactic or morphological changes. Comparisons of words from different languages are traditionally carried out by hand, introducing an element of subjectivity into diachronic linguistics. Early attempts to quantify the similarity between languages (e.g., [15]) made drastic simplifying assumptions that drew strong criticism from diachronic linguists. In particular, many of these approaches simply represent the appearance of a word in two languages with a single bit, rather than allowing for gradations based on correspondences between sequences of phonemes.

We take a quantitative approach to diachronic linguistics that alleviates this problem by operating at the phoneme level. Our approach combines the advantages of the classical, phoneme-based, comparative method with the robustness of corpus-based probabilistic models. We focus on the case where the words are etymological cognates across languages, e.g. French *faire* and Spanish *hacer* from Latin *facere* (to do). Following [3], we use this information to estimate a contextualized model of phonological change expressed as a probability distribution over rules applied to individual phonemes. The model is fully generative, and thus can be used to solve a variety of problems. For example, we can reconstruct ancestral word forms or inspect the rules learned along each branch of

a phylogeny to identify sound laws. Alternatively, we can observe a word in one or more modern languages, say French and Spanish, and query the corresponding word form in another language, say Italian. Finally, models of this kind can potentially be used as a building block in a system for inferring the topology of phylogenetic trees [3].

In this paper, we use this general approach to evaluate the performance of two different schemes for defining probability distributions over rules. The first scheme, used in [3], treats these distributions as simple multinomials and uses a Dirichlet prior on these multinomials. This approach makes it difficult to capture rules that apply at different levels of granularity. Inspired by the prevalence of multi-scale rules in diachronic phonology and modern phonological theory, we develop a new scheme in which rules possess a set of features, and a distribution over rules is defined using a log-linear model. We evaluate both schemes in reconstructing ancient word forms, showing that the new linguistically-motivated change can improve performance significantly.

## 2   Background and previous work

Most previous computational approaches to diachronic linguistics have focused on the reconstruction of phylogenetic trees from a Boolean matrix indicating the properties of words in different languages [10, 6, 14, 13]. These approaches descend from *glottochronology* [15], which measures the similarity between languages (and the time since they diverged) using the number of words in those languages that belong to the same cognate set. This information is obtained from manually curated cognate lists such as the data of [5]. The modern instantiations of this approach rely on sophisticated techniques for inferring phylogenies borrowed from evolutionary biology (e.g., [11, 7]). However, they still generally use cognate sets as the basic data for evaluating the similarity between languages (although some approaches incorporate additional manually constructed features [14]).

As an example of a cognate set encoding, consider the meaning "eat". There would be one column for the cognate set which appears in French as *manger* and Italian as *mangiare* since both descend from the Latin *mandere* (to chew). There would be another column for the cognate set which appears in both Spanish and Portuguese as *comer*, descending from the Latin *comedere* (to consume). If these were the only data, algorithms based on this data would tend to conclude that French and Italian were closely related and that Spanish and Portuguese were equally related. However, the cognate set representation has several disadvantages: it does not capture the fact that the cognate is closer between Spanish and Portuguese than between French and Spanish, nor do the resulting models let us conclude anything about the regular processes which caused these languages to diverge. Also, curating cognate data can be expensive. In contrast, each word in our work is tracked using an automatically obtained cognate list. While these cognates may be noisier, we compensate for this by modeling phonological changes rather than Boolean mutations in cognate sets.

Another line of computational work has explored using phonological models as a way to capture the differences between languages. [16] describes an information theoretic measure of the distance between two dialects of Chinese. They use a probabilistic edit model, but do not consider the reconstruction of ancient word forms, nor do they present a learning algorithm for such models. There have also been several approaches to the problem of cognate prediction in machine translation (essentially transliteration), e.g., [12]. Compared to our work, the phenomena of interest, and therefore the models, are different. [12] presents a model for learning "sound laws," general phonological changes governing two completely observed aligned cognate lists. This model can be viewed as a special case of ours using a simple two-node topology.

## 3   A generative model of phonological change

In this section, we outline the framework for modeling phonological change that we will use throughout the paper. Assume we have a fixed set of *word types* (cognate sets) in our vocabulary $V$ and a set of languages $L$. Each word type $i$ has a *word form* $w_{il}$ in each language $l \in L$, which is represented as a sequence of phonemes which might or might not be observed. The languages are arranged according to some tree topology $T$ (see Figure 2(a) for examples). It is possible to also induce the topology or cognate set assignments, but in this paper we assume that the topology is fixed and cognates have already been identified.

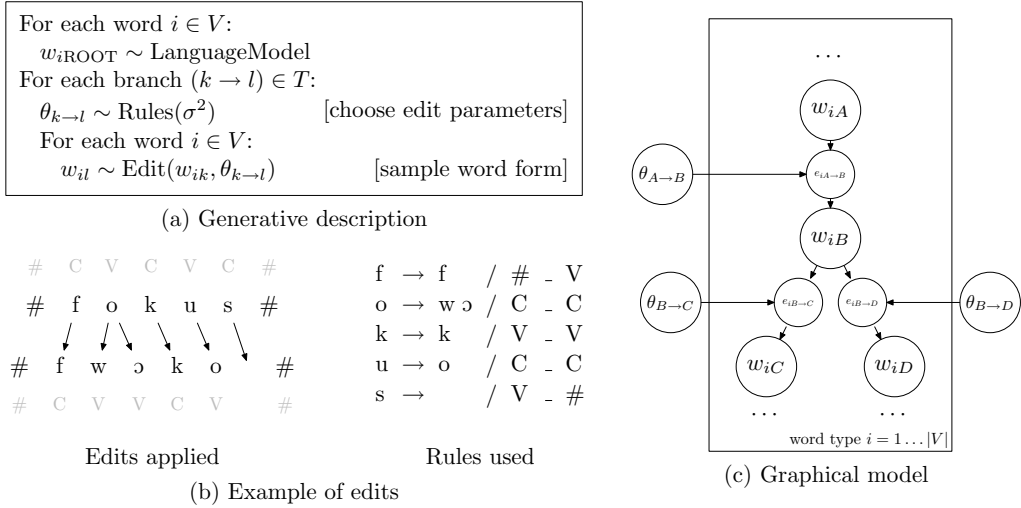

For each word $i \in V$:
$\quad w_{i\text{ROOT}} \sim \text{LanguageModel}$
For each branch $(k \to l) \in T$:
$\quad \theta_{k \to l} \sim \text{Rules}(\sigma^2)$       [choose edit parameters]
$\quad$ For each word $i \in V$:
$\quad\quad w_{il} \sim \text{Edit}(w_{ik}, \theta_{k \to l})$       [sample word form]

(a) Generative description

| | | | | | |
|#|C|V|C|V|C|#|
|#|f|o|k|u|s|#|

| | | | | | |
|#|f|w|ɔ|k|o| |#|
|#|C|V|V|C|V| |#|

Edits applied

f → f / # _ V
o → w ɔ / C _ C
k → k / V _ V
u → o / C _ C
s → / V _ #

Rules used

(b) Example of edits

(c) Graphical model

Figure 1: (a) A description of the generative model. (b) An example of edits that were used to transform the Latin word *focus* (*/fokus/*) into the Italian word *fuoco* (/fwɔko/) (fire) along with the context-specific rules that were applied. (c) The graphical model representation of our model: $\theta$ are the parameters specifying the stochastic edits $e$, which govern how the words $w$ evolve.

The probabilistic model specifies a distribution over the word forms $\{w_{il}\}$ for each word type $i \in V$ and each language $l \in L$ via a simple generative process (Figure 1(a)). The generative process starts at the root language and generates all the word forms in each language in a top-down manner. The $w \sim \text{LanguageModel}$ distribution is a simple bigram phoneme model. A root word form $w$ consisting of $n$ phonemes $x_1 \cdots x_n$ is generated with probability $p_{\text{lm}}(x_1) = \prod_{j=2}^{n} p_{\text{lm}}(x_j \mid x_{j-1})$, where $p_{\text{lm}}$ is the distribution of the language model. The stochastic edit model $w' \sim \text{Edit}(w, \theta)$ describes how a single old word form $w = x_1 \cdots x_n$ changes along one branch of the phylogeny with parameters $\theta$ to produce a new word form $w'$. This process is parametrized by rule probabilities $\theta_{k \to l}$, which are specific to branch $(k \to l)$.

The generative process used in the edit model is as follows: for each phoneme $x_i$ in the old word form, walking from left to right, choose a rule to apply. There are three types of rules: (1) *deletion* of the phoneme, (2) *substitution* with some phoneme (possibly the same one), or (3) *insertion* of another phoneme, either before or after the existing one. The probability of applying a rule depends on the *context* $(x_{i-1}, x_{i+1})$. Context-dependent rules are often used to characterize phonological changes in diachronic linguistics [4]. Figure 1(b) shows an example of the rules being applied. The context-dependent form of these rules allows us to represent phenomena such as the likely deletion of $s$ in word-final positions.

## 4 Defining distributions over rules

In the model defined in the previous section, each branch $(k \to l) \in T$ has a collection of context-dependent rule probabilities $\theta_{k \to l}$. Specifically, $\theta_{k \to l}$ specifies a collection of multinomial distributions, one for each $C = (c_l, x, c_r)$, where $c_l$ is left phoneme, $x$ is the old phoneme, $c_r$ is the right phoneme. Each multinomial distribution is over possible right-hand sides $\alpha$ of the rule, which could consist of 0, 1, or 2 phonemes. We write $\theta_{k \to l}(C, \alpha)$ for the probability of rule $x \to \alpha / c_1 \_ c_2$.

Previous work using this probabilistic framework simply placed independent Dirichlet priors on each of the multinomial distributions [3]. While this choice results in a simple estimation procedure, it has some severe limitations. Sound changes happen at many granularities. For example, from Latin to Vulgar Latin, $u \to o$ occurs in many contexts while $s \to \emptyset$ occurs only in word-final contexts. Using independent Dirichlets forces us to commit to a single context granularity for $C$. Since the different multinomial distributions are not tied together, generalization becomes very difficult, especially as data is limited. It is also difficult to interpret the learned rules, since the evidence for a coarse phenomenon such as $u \to o$ would be unnecessarily fragmented across many different

context-dependent rules. We would like to ideally capture a phenomenon using a single rule or feature. We could relate the rule probabilities via a simple hierarchical Bayesian model, but we would still have to define a single hierarchy of contexts. This restriction might be inappropriate given that sound changes often depend on different contexts that are not necessarily nested.

For these reasons, we propose using a feature-based distribution over the rule probabilities. Let $F(C, \alpha)$ be a feature vector that depends on the context-dependent rule $(C, \alpha)$, and $\lambda_{k \to l}$ be the log-linear weights for branch $(k \to l)$. We use a Normal prior on the log-linear weights, $\lambda_{k \to l} \sim \mathcal{N}(0, \sigma^2 I)$. The rule probabilities are then deterministically related to the weights via the softmax function:

$$\theta_{k \to l}(C, \alpha; \lambda_{k \to l}) = \frac{e^{\lambda_{k \to l}^T F(C, \alpha)}}{\sum_{\alpha'} e^{\lambda_{k \to l}^T F(C, \alpha')}}. \tag{1}$$

For each rule $x \to \alpha \ / \ c_l \ _- \ c_r$, we defined features based on whether $x = \alpha$ (i.e. self-substitution), and whether $|\alpha| = n$ for each $n = 0, 1, 2$ (corresponding to deletion, substitution, and insertion). We also defined sets of features using three partitions of phonemes $c$ into "natural classes". These correspond to looking at the place of articulation (denoted $A_2(c)$), testing whether $c$ is a vowel, consonant, or boundary symbol ($A_1(c)$), and the trivial wildcard partition ($A_0(c)$), which allows rules to be insensitive to $c$. Using these partitions, the final set of features corresponded to whether $A_{k_l}(c_l) = a_l$ and $A_{k_r}(c_r) = a_r$ for each type of partitioning $k_l, k_r \in \{0, 1, 2\}$ and natural classes $a_l, a_r$.

The move towards using a feature-based scheme for defining rule probabilities is not just motivated by the greater expressive capacity of this scheme. It also provides a connection with contemporary phonological theory. Recent work in computational linguistics on probabilistic forms of optimality theory has begun to use a similar approach, characterizing the distribution over word forms within a language using a log-linear model applied to features of the words [17, 9]. Using similar features to define a distribution over phonological changes thus provides a connection between synchronic and diachronic linguistics in addition to a linguistically-motivated method for improving reconstruction.

## 5    Learning and inference

We use a Monte Carlo EM algorithm to fit the parameters of both models. The algorithm iterates between a stochastic E-step, which computes reconstructions based on the current edit parameters, and an M-step, which updates the edit parameters based on the reconstructions.

### 5.1    Monte Carlo E-step: sampling the edits

The E-step computes the expected sufficient statistics required for the M-step, which in our case is the expected number of times each edit (such as o $\to$ ɔ) was used in each context. Note that the sufficient statistics do not depend on the prior over rule probabilities; in particular, both the model based on independent Dirichlet priors and the one based on a log-linear prior require the same E-step computation.

An exact E-step would require summing over all possible edits involving all languages in the phylogeny (all unobserved $\{e\}, \{w\}$ variables in Figure 1(c)), which does not permit a tractable dynamic program. Therefore, we resort to a Monte Carlo E-step, where many samples of the edit variables are collected, and counts are computed based on these samples. Samples are drawn using Gibbs sampling [8]: for each word form of a particular language $w_{il}$, we fix all other variables in the model and sample $w_{il}$ along with its corresponding edits.

Consider the simple four-language topology in Figure 1(c). Suppose that the words in languages $A$, $C$ and $D$ are fixed, and we wish to sample the word at language $B$ along with the three corresponding sets of edits (remember that the edits fully determine the words). While there are an exponential number of possible words/edits, we can exploit the Markov structure in the edit model to consider all such words/edits using dynamic programming, in a way broadly similar to the forward-backward algorithm for HMMs. See [3] for details of the dynamic program.

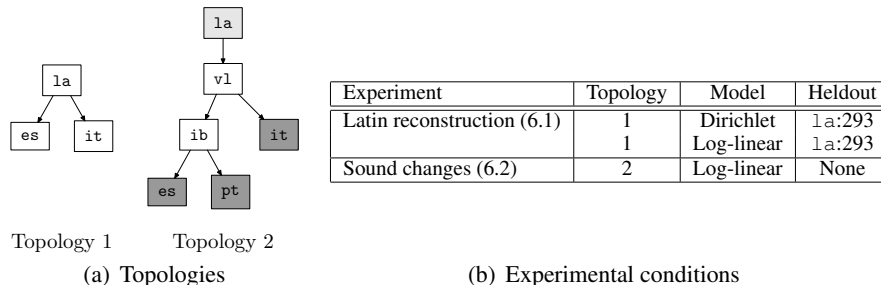

Topology 1      Topology 2

(a) Topologies            (b) Experimental conditions

| Experiment | Topology | Model | Heldout |
|---|---|---|---|
| Latin reconstruction (6.1) | 1 | Dirichlet | la:293 |
| | 1 | Log-linear | la:293 |
| Sound changes (6.2) | 2 | Log-linear | None |

Figure 2: Conditions under which each of the experiments presented in this section were performed. The *topology* indices correspond to those displayed at the left. The heldout column indicates how many words, if any, were held out for edit distance evaluation, and from which language. All the experiments were run on a data set of 582 cognates from [3].

## 5.2 M-step: updating the parameters

In the M-step, we estimate the distribution over rules for each branch $(k \to l)$. In the Dirichlet model, this can be done in closed form [3]. In the log-linear model, we need to optimize the feature weights $\lambda_{k \to l}$. Let us fix a single branch and drop the subscript. Let $N(C, \alpha)$ be the expected number of times the rule $(C, \alpha)$ was used in the E-step. Given these sufficient statistics, the estimate of $\lambda$ is given by optimizing the expected complete log-likelihood plus the regularization penalty from the prior on $\lambda$,

$$\mathcal{O}(\lambda) = \sum_{C,\alpha} N(C, \alpha) \Big[ \lambda^T F(C, \alpha) - \log \sum_{\alpha'} e^{\lambda^T F(C, \alpha')} \Big] - \frac{||\lambda||^2}{2\sigma^2}. \tag{2}$$

We use L-BFGS to optimize this convex objective. which only requires the partial derivatives:

$$\frac{\partial \mathcal{O}(\lambda)}{\partial \lambda_j} = \sum_{C,\alpha} N(C, \alpha) \Big[ F_j(C, \alpha) - \sum_{\alpha'} \theta(C, \alpha'; \lambda) F_j(C, \alpha') \Big] - \frac{\lambda_j}{\sigma^2} \tag{3}$$

$$= \hat{F}_j - \sum_{C,\alpha'} N(C, \cdot) \theta(C, \alpha'; \lambda) F_j(C, \alpha') - \frac{\lambda_j}{\sigma^2}, \tag{4}$$

where $\hat{F}_j \stackrel{\text{def}}{=} \sum_{C,\alpha} N(C, \alpha) F_j(C, \alpha)$ is the empirical feature vector and $N(C, \cdot) \stackrel{\text{def}}{=} \sum_{\alpha} N(C, \alpha)$ is the number of times context $C$ was used. $\hat{F}_j$ and $N(C, \cdot)$ do not depend on $\lambda$ and thus can be precomputed at the beginning of the M-step, thereby speeding up each L-BFGS iteration.

## 6 Experiments

In this section, we summarize the results of the experiments testing our different probabilistic models of phonological change. The experimental conditions are summarized in Table 2. Training and test data sets were taken from [3].

### 6.1 Reconstruction of ancient word forms

We ran the two models using Topology 1 in Figure 2 to assess the relative performance of Dirichlet-parametrized versus log-linear-parametrized models. Half of the Latin words at the root of the tree were held out, and the (uniform cost) Levenshtein edit distance from the predicted reconstruction to the truth was computed. While the uniform-cost edit distance misses important aspects of phonology (all phoneme substitutions are not equal, for instance), it is parameter-free and still seems to correlate to a large extent with linguistic quality of reconstruction. It is also superior to held-out log-likelihood, which fails to penalize errors in the modeling assumptions, and to measuring the percentage of perfect reconstructions, which ignores the degree of correctness of each reconstructed word.

| Model | Baseline | Model | Improvement |
|---|---|---|---|
| Dirichlet | 3.59 | 3.33 | 7% |
| Log-linear (0) | 3.59 | 3.21 | 11% |
| Log-linear (0,1) | 3.59 | 3.14 | 12% |
| Log-linear (0,1,2) | 3.59 | 3.10 | 14% |

Table 1: Results of the edit distance experiment. The *language* column corresponds to the language held out for evaluation. We show the mean edit distance across the evaluation examples. *Improvement rate* is computed by comparing the score of the algorithm against the baseline described in Section 6.1. The numbers in parentheses for the log-linear model indicate which levels of granularity were used to construct the features (see Section 4).

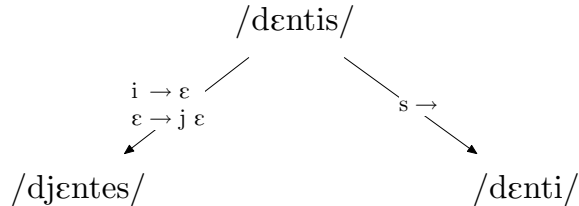

Figure 3: An example of the proper Latin reconstruction given the Spanish and Italian word forms. Our model produces /dɛntes/, which is nearly correct, capturing two out of three of the phenomena.

We ran EM for 10 iterations for each model, and evaluated performance via a Viterbi derivation produced using these parameters. Our baseline for comparison was picking randomly, for each heldout node in the tree, an observed neighboring word (i.e., copy one of the modern forms). Both models outperformed this baseline (see Figure 3), and the log-linear model outperformed the Dirichlet model, suggesting that the featurized system better captures the phonological changes. Moreover, adding more features further improved the performance, indicating that being able to express rules at multiple levels of granularity allows the model to capture the underlying phonological changes more accurately.

To give a qualitative feel for the operation of the system (good and bad), consider the example in Figure 3, taken from the Dirichlet-parametrized experiment. The Latin *dentis* /dɛntis/ (teeth) is nearly correctly reconstructed as /dɛntes/, reconciling the appearance of the /j/ in the Spanish and the disappearance of the final /s/ in the Italian. Note that the /is/ vs. /es/ ending is difficult to predict in this context (indeed, it was one of the early distinctions to be eroded in Vulgar Latin).

## 6.2 Inference of phonological changes

Another use of this model is to automatically recover the phonological drift processes between known or partially-known languages. To facilitate evaluation, we continued in the well-studied Romance evolutionary tree. Again, the root is Latin, but we now add an additional modern language, Portuguese, and two additional hidden nodes. One of the nodes characterizes the least common ancestor of modern Spanish and Portuguese; the other, the least common ancestor of all three modern languages. In Figure 2, Topology 2, these two nodes are labeled vl (Vulgar Latin) and ib (Proto-Ibero Romance), respectively. Since we are omitting many other branches, these names should not be understood as referring to actual historical proto-languages, but, at best, to collapsed points representing several centuries of evolution. Nonetheless, the major reconstructed rules still correspond to well-known phenomena and the learned model generally places them on reasonable branches.

Figure 4 shows the top four general rules for each of the evolutionary branches recovered by the log-linear model. The rules are ranked by the number of times they were used in the derivations during the last iteration of EM. The la, es, pt, and it forms are fully observed while the vl and ib forms are automatically reconstructed. Figure 4 also shows a specific example of the evolution of the Latin *VERBUM* (word), along with the specific edits employed by the model.

For this particular example, both the Dirichlet and the log-linear models produced the same reconstruction in the internal nodes. However, the log-linear parametrization makes inspection of sound laws easier. Indeed, with the Dirichlet model, since the natural classes are of fixed granularity, some

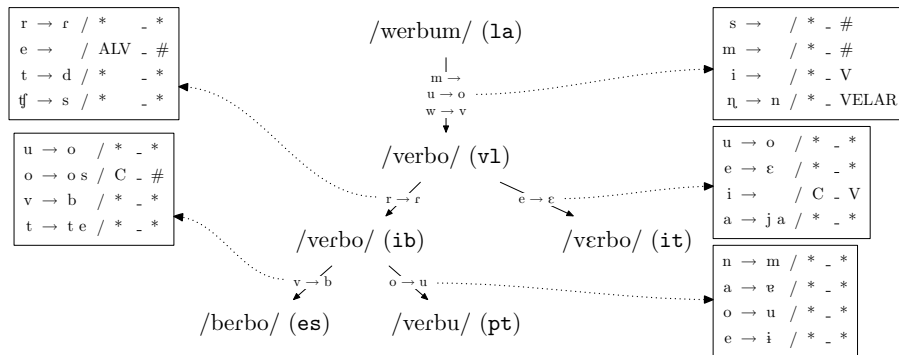

Figure 4: The tree shows the system's hypothesized transformation of a selected Latin word form, *VERBUM* (word) into the modern Spanish, Italian, and Portuguese pronunciations. The Latin root and modern leaves were observed while the hidden nodes as well as all the derivations were obtained using the parameters computed by our model after 10 iterations of EM. Nontrivial rules (i.e. rules that are not identities) used at each stage are shown along the corresponding edge. The boxes display the top four nontrivial rules corresponding to each of these evolutionary branches, ordered by the number of times they were applied during the last E step. These are grouped and labeled by their active feature of highest weight. ALV stands for alveolar consonant.

rules must be redundantly discovered, which tends to flood the top of the rule lists with duplicates. In contrast, the log-linear model groups rules with features of the appropriate degree of generality.

While quantitative evaluation such as measuring edit distance is helpful for comparing results, it is also illuminating to consider the plausibility of the learned parameters in a historical light, which we do here briefly. In particular, we consider rules on the branch between `la` and `vl`, for which we have historical evidence. For example, documents such as the *Appendix Probi* [2] provide indications of orthographic confusions which resulted from the growing gap between Classical Latin and Vulgar Latin phonology around the 3rd and 4th centuries AD. The *Appendix* lists common misspellings of Latin words, from which phonological changes can be inferred.

On the `la` to `vl` branch, rules for word-final deletion of classical case markers dominate the list. It is indeed likely that these were generally eliminated in Vulgar Latin. For the deletion of the /m/, the *Appendix Probi* contains pairs such as *PASSIM NON PASSI* and *OLIM NON OLI*. For the deletion of final /s/, this was observed in early inscriptions, e.g. *CORNELIO* for *CORNELIOS* [1]. The frequent leveling of the distinction between /o/ and /u/ (which was ranked 5, but was not included for space reasons) can be also be found in the *Appendix Probi*: *COLUBER NON COLOBER*. Note that in the specific example shown, the model lowers the original /u/ and then re-raises it in the `pt` branch due to a later process along that branch.

Similarly, major canonical rules were discovered in other branches as well, for example, /v/ to /b/ fortition in Spanish, palatalization along several branches, and so on. Of course, the recovered words and rules are not perfect. For example, reconstructed Ibero /trinta/ to Spanish /treinta/ (thirty) is generated in an odd fashion using rules /e/ to /i/ and /n/ to /in/. In the Dirichlet model, even when otherwise reasonable systematic sound changes are captured, the crudeness of the fixed-granularity contexts can prevent the true context from being captured, resulting in either rules applying with low probability in overly coarse environments or rules being learned redundantly in overly fine environments. The featurized model alleviates this problem.

## 7 Conclusion

Probabilistic models have the potential to replace traditional methods used for comparing languages in diachronic linguistics with quantitative methods for reconstructing word forms and inferring phylogenies. In this paper, we presented a novel probabilistic model of phonological change, in which the rules governing changes in the sound of words are parametrized using the features of the phonemes involved. This model goes beyond previous work in this area, providing more accurate reconstructions of ancient word forms and connections to current work on phonology in synchronic linguistics. Using a log-linear model to define the probability of a rule being applied results in a

straightforward inference procedure which can be used to both produce accurate reconstructions as measured by edit distance and identify linguistically plausible rules that account for phonological changes. We believe that this probabilistic approach has the potential to support quantitative analysis of the history of languages in a way that can scale to large datasets while remaining sensitive to the concerns that have traditionally motivated diachronic linguistics.

**Acknowledgments**    We would like to thank Bonnie Chantarotwong for her help with the IPA converter and our reviewers for their comments. This work was supported by a FQRNT fellowship to the first author, a NDSEG fellowship to the second author, NSF grant number BCS-0631518 to the third author, and a Microsoft Research New Faculty Fellowship to the fourth author.

# References

[1] W. Sidney Allen. *Vox Latina: The Pronunciation of Classical Latin*. Cambridge University Press, 1989.

[2] W.A. Baehrens. *Sprachlicher Kommentar zur vulgärlateinischen Appendix Probi*. Halle (Saale) M. Niemeyer, 1922.

[3] A. Bouchard-Côté, P. Liang, T. Griffiths, and D. Klein. A Probabilistic Approach to Diachronic Phonology. In *Empirical Methods in Natural Language Processing and Computational Natural Language Learning (EMNLP/CoNLL)*, 2007.

[4] L. Campbell. *Historical Linguistics*. The MIT Press, 1998.

[5] I. Dyen, J.B. Kruskal, and P. Black. FILE IE-DATA1. Available at http://www.ntu.edu.au/education/langs/ielex/IE-DATA1, 1997.

[6] S. N. Evans, D. Ringe, and T. Warnow. Inference of divergence times as a statistical inverse problem. In P. Forster and C. Renfrew, editors, *Phylogenetic Methods and the Prehistory of Languages*. McDonald Institute Monographs, 2004.

[7] J. Felsenstein. *Inferring Phylogenies*. Sinauer Associates, 2003.

[8] S. Geman and D. Geman. Stochastic relaxation, Gibbs distributions, and the Bayesian restoration of images. *IEEE Transactions on Pattern Analysis and Machine Intelligence*, 6:721–741, 1984.

[9] S. Goldwater and M. Johnson. Learning ot constraint rankings using a maximum entropy model. *Proceedings of the Workshop on Variation within Optimality Theory*, 2003.

[10] R. D. Gray and Q. Atkinson. Language-tree divergence times support the Anatolian theory of Indo-European origins. *Nature*, 2003.

[11] J. P. Huelsenbeck, F. Ronquist, R. Nielsen, and J. P. Bollback. Bayesian inference of phylogeny and its impact on evolutionary biology. *Science*, 2001.

[12] G. Kondrak. *Algorithms for Language Reconstruction*. PhD thesis, University of Toronto, 2002.

[13] L. Nakhleh, D. Ringe, and T. Warnow. Perfect phylogenetic networks: A new methodology for reconstructing the evolutionary history of natural languages. *Language*, 81:382–420, 2005.

[14] D. Ringe, T. Warnow, and A. Taylor. Indo-european and computational cladistics. *Transactions of the Philological Society*, 100:59–129, 2002.

[15] M. Swadesh. Towards greater accuracy in lexicostatistic dating. *Journal of American Linguistics*, 21:121–137, 1955.

[16] A. Venkataraman, J. Newman, and J.D. Patrick. A complexity measure for diachronic chinese phonology. In J. Coleman, editor, *Computational Phonology*. Association for Computational Linguistics, 1997.

[17] C. Wilson and B. Hayes. A maximum entropy model of phonotactics and phonotactic learning. *Linguistic Inquiry*, 2007.

